# The Geometry of Eye Rotations and Listing's Law

**Amir A. Handzel***         **Tamar Flash[†]**
Department of Applied Mathematics and Computer Science
Weizmann Institute of Science
Rehovot, 76100 Israel

## Abstract

We analyse the geometry of eye rotations, and in particular saccades, using basic Lie group theory and differential geometry. Various parameterizations of rotations are related through a unifying mathematical treatment, and transformations between co-ordinate systems are computed using the Campbell–Baker–Hausdorff formula. Next, we describe Listing's law by means of the Lie algebra $so(3)$. This enables us to demonstrate a direct connection to Donders' law, by showing that eye orientations are restricted to the quotient space $SO(3)/SO(2)$. The latter is equivalent to the sphere $\mathcal{S}^2$, which is exactly the space of gaze directions. Our analysis provides a mathematical framework for studying the oculomotor system and could also be extended to investigate the geometry of multi-joint arm movements.

## 1 INTRODUCTION

### 1.1 SACCADES AND LISTING'S LAW

Saccades are fast eye movements, bringing objects of interest into the center of the visual field. It is known that eye positions are restricted to a subset of those which are anatomically possible, both during saccades and fixation (Tweed & Vilis, 1990). According to Donders' law, the eye's gaze direction determines its orientation uniquely, and moreover, the orientation does not depend on the history of eye motion which has led to the given gaze direction. A precise specification of the "allowed" subspace of position is given by Listing's law: the observed orientations of the eye are those which can be reached from the distinguished orientation called *primary*

[†]tamar@wisdom.weizmann.ac.il

*position* through a single rotation about an axis which lies in the plane perpendicular to the gaze direction at the primary position (Listing's plane). We say then that the orientation of the eye has zero torsion. Recently, the domain of validity of Listing's law has been extended to include eye vergence by employing a suitable mathematical treatment (Van Rijn & Van Den Berg, 1993).

Tweed and Vilis used quaternion calculus to demonstrate, in addition, that in order to move from one allowed position to another in a single rotation, the rotation axis itself lies outside Listing's plane (Tweed & Vilis, 1987). Indeed, normal saccades are performed approximately about a single axis. However, the validity of Listing's law does not depend on the rotation having a single axis, as was shown in double-step target displacement experiments (Minken, Van Opstal & Van Gisbergen, 1993): even when the axis of rotation itself changes during the saccade, Listing's law is obeyed at each and every point along the trajectory which is traced by the eye.

Previous analyses of eye rotations (and in particular of Listing's law) have been based on various representations of rotations: quaternions (Westheimer, 1957), rotation vectors (Hepp, 1990), spinors (Hestenes, 1994) and $3 \times 3$ rotation matrices; however, they are all related through the same underlying mathematical object — the three dimensional (3D) rotation group. In this work we analyse the geometry of saccades using the Lie algebra of the rotation group and the group structure. Next, we briefly describe the basic mathematical notions which will be needed later. This is followed by Section 2 in which we analyse various parameterizations of rotations from the point of view of group theory; Section 3 contains a detailed mathematical analysis of Listing's law and its connection to Donders' law based on the group structure; in Section 4 we briefly discuss the issue of angular velocity vectors or axes of rotation ending with a short conclusion.

## 1.2   THE ROTATION GROUP AND ITS LIE ALGEBRA

The group of rotations in three dimensions, $G = SO(3)$, (where '$SO$' stands for special orthogonal transformations) is used both to describe actual rotations and to denote eye positions by means of a unique virtual rotation from the primary position. The identity operation leaves the eye at the primary position, therefore, we identify this position with the unit element of the group $e \in SO(3)$. A rotation can be parameterized by a 3D axis and the angle of rotation about it. Each axis "generates" a continuous set of rotations through increasing angles. Formally, if $n$ is a unit axis of rotation, then

$$\text{EXP}(\boldsymbol{\theta} \cdot \boldsymbol{n}) \tag{1}$$

is a continuous one-parameter subgroup (in $G$) of rotations through angles $\theta$ in the plane that is perpendicular to $\boldsymbol{n}$. Such a subgroup is denoted as $SO(2) \subset SO(3)$. We can take an explicit representation of $\boldsymbol{n}$ as a matrix and the exponent can be calculated as a Taylor series expansion. Let us look, for example, at the one parameter subgroup of rotations in the $y-z$ plane, i.e. rotations about the $x$ axis which is represented in this case by the matrix

$$L_x = \begin{pmatrix} 0 & 0 & 0 \\ 0 & 0 & 1 \\ 0 & -1 & 0 \end{pmatrix}. \tag{2}$$

A direct computation of this rotation by an angle $\theta$ gives

$$\text{EXP}(\theta L_x) = I + \theta L_x + \frac{1}{2!}(\theta L_x)^2 + \ldots + \frac{1}{n!}(\theta L_x)^n + \ldots = \begin{pmatrix} 1 & 0 & 0 \\ 0 & \cos\theta & \sin\theta \\ 0 & -\sin\theta & \cos\theta \end{pmatrix} \tag{3}$$

where $I$ is the identity matrix. Thus, the rotation matrix $R(\theta)$ can be constructed from the axis and angle of rotation. The same rotation, however, could also be achieved using $\lambda L_x$ instead of $L_x$, where $\lambda$ is any scalar, while rescaling the angle to $\theta/\lambda$. The collection of matrices $\lambda L_x$ is a one dimensional linear space whose elements are the *generators* of rotations in the $y-z$ plane.

The set of all the generators constitutes the *Lie algebra* of a group. For the full space of 3D rotations, the Lie algebra is the three dimensional vector space that is spanned by the standard orthonormal basis comprising the three direction vectors of the principal axes:

$$g = so(3) = Span\{\mathbf{e}_x, \mathbf{e}_y, \mathbf{e}_z\}. \tag{4}$$

Every axis $n$ can be expressed as a linear combination of this basis. Elements of the Lie algebra can also be represented in matrix form and the corresponding basis for the matrix space is

$$L_x = \begin{pmatrix} 0 & 0 & 0 \\ 0 & 0 & 1 \\ 0 & -1 & 0 \end{pmatrix} L_y = \begin{pmatrix} 0 & 0 & 1 \\ 0 & 0 & 0 \\ -1 & 0 & 0 \end{pmatrix} L_z = \begin{pmatrix} 0 & 1 & 0 \\ -1 & 0 & 0 \\ 0 & 0 & 0 \end{pmatrix}; \tag{5}$$

hence we have the isomorphism

$$\begin{pmatrix} 0 & \theta_z & \theta_y \\ -\theta_z & 0 & \theta_x \\ -\theta_y & -\theta_x & 0 \end{pmatrix} \longleftrightarrow \begin{pmatrix} \theta_x \\ \theta_y \\ \theta_z \end{pmatrix}. \tag{6}$$

Thanks to its linear structure, the Lie algebra is often more convenient for analysis than the group itself. In addition to the linear structure, the Lie algebra has a bilinear antisymmetric operation defined between its elements which is called the *bracket* or *commutator*. The bracket operation between vectors in $g$ is the usual vector cross product. When the elements of the Lie algebra are written as matrices, the bracket operation becomes a commutation relation, i.e.

$$[A, B] \equiv AB - BA. \tag{7}$$

As expected, the commutation relations of the basis matrices of the Lie algebra (of the 3D rotation group) are equivalent to the vector product:

$$[L_i, L_j] = \epsilon_{ijk} L_k \tag{8}$$

Finally, in accordance with (1), every rotation matrix is obtained by exponentiation:

$$R(\boldsymbol{\theta}) = \text{EXP}(\theta_x L_x + \theta_y L_y + \theta_z L_z). \tag{9}$$

where $\boldsymbol{\theta}$ stands for the three component angles.

## 2    CO-ORDINATE SYSTEMS FOR ROTATIONS

In linear spaces the "position" of a point is simply parameterized by the co-ordinates w.r.t. the principal axes (a chosen orthonormal basis). For a non-linear space (such as the rotation group) we define local co-ordinate charts that look like pieces of a vector space $\mathbb{R}^n$. Several co-ordinate systems for rotations are based on the fact that group elements can be written as exponents of elements of the Lie algebra (1). The angles $\boldsymbol{\theta}$ appearing in the exponent serve as the co-ordinates. The underlying property which is essential for comparing these systems is the non-commutativity of rotations. For usual real numbers, e.g. $c_1$ and $c_2$, commutativity implies $\exp^{c_1} \exp^{c_2} = \exp^{c_1+c_2}$. A corresponding equation for non-commuting elements is the Campbell-Baker-Hausdorff formula (CBH) which is a Taylor series

expansion using repeated commutators between the elements of the Lie algebra. The expansion to third order is (Choquet-Bruhat et al., 1982):

$$\text{EXP}(x_1)\text{EXP}(x_2) = \text{EXP}\left(x_1 + x_2 + \frac{1}{2}[x_1, x_2] + \frac{1}{12}[x_1 - x_2, [x_1, x_2]]\right) \quad (10)$$

where $x_1, x_2$ are variables that stand for elements of the Lie algebra.

One natural parameterization uses the representation of a rotation by the axis and the angle of rotation. The angles which appear in (9) are then called *canonical co-ordinates of the first kind* (Varadarajan, 1974). Gimbal systems constitute a second type of parameterization where the overall rotation is obtained by a series of consecutive rotations about the principal axes. The component angles are then called *canonical co-ordinates of the second kind*. In the present context, the first type of co-ordinates are advantageous because they correspond to single axis rotations which in turn represent natural eye movements. For convenience, we will use the name *canonical* co-ordinates for those of the first kind, whereas those of the second type will simply be called gimbals. The gimbals of Fick and Helmholtz are commonly used in the study of oculomotor control (Van Opstal, 1993). A rotation matrix in Fick gimbals is

$$R_F(\theta_x, \theta_y, \theta_z) = \text{EXP}(\theta_z L_z) \cdot \text{EXP}(\theta_y L_y) \cdot \text{EXP}(\theta_x L_x), \quad (11)$$

and in Helmholtz gimbals the order of rotations is different:

$$R_H(\theta_x, \theta_y, \theta_z) = \text{EXP}(\theta_y L_y) \cdot \text{EXP}(\theta_z L_z) \cdot \text{EXP}(\theta_x L_x). \quad (12)$$

The CBH formula (10) can be used as a general tool for obtaining transformations between various co-ordinate systems (Gilmore, 1974) such as (9,11,12). In particular, we apply (10) to the product of the two right-most terms in (11) and then again to the product of the result with the third term. We thus arrive at an expression whose form is the same as the right hand side of (10). By equating it with the expression for canonical angles (9) and then taking the *log* of the exponents on both sides of the equation, we obtain the transformation formula from Fick angles to canonical angles. Repeating this calculation for (12) gives the equivalent formula for Helmholtz angles[1]. Both transformations are given by the following three equations where $\theta^{F,H}$ stands for an angle either in Fick or in Helmholtz co-ordinates; for Helmholtz angles there is a plus sign in front of the last term of the first equation and a minus sign in the case of Fick angles:

$$\theta_x^C = \theta_x^{F,H}\left(1 - \frac{1}{12}\left((\theta_y^{F,H})^2 + (\theta_z^{F,H})^2\right)\right) \pm \frac{1}{2}\theta_y^{F,H}\theta_z^{F,H}$$

$$\theta_y^C = \theta_y^{F,H}\left(1 - \frac{1}{12}\left((\theta_z^{F,H})^2 + (\theta_x^{F,H})^2\right)\right) + \frac{1}{2}\theta_z^{F,H}\theta_x^{F,H} \quad (13)$$

$$\theta_z^C = \theta_z^{F,H}\left(1 - \frac{1}{12}\left((\theta_x^{F,H})^2 + (\theta_y^{F,H})^2\right)\right) - \frac{1}{2}\theta_x^{F,H}\theta_y^{F,H}$$

The error caused by the above approximation is smaller than 0.1 degree within most of the oculomotor range.

We mention in closing two additional parameterizations, namely quaternions and rotation vectors. Unit quaternions lie on the 3D sphere $\mathcal{S}^3$ (embedded in $\mathbb{R}^4$) which constitutes the same manifold as the group of unitary rotations $SU(2)$. The latter is the double covering group of $SO(3)$ having the same local structure. This enables to use quaternions to parameterize rotations. The popular rotation vectors (written as $\tan(\theta/2)n$, $n$ being the axis of rotation and $\theta$ its angle) are closely related to

quaternions because they are central (gnomonic) projections of a hemisphere of $\mathcal{S}^3$ onto the 3D affine space tangent to the quaternion $\boldsymbol{q}_e = (1, 0, 0, 0) \in \mathbb{R}^4$. [2]

## 3   LISTING'S LAW AND DONDERS' LAW

A customary choice of a head fixed coordinate system is the following: $\mathbf{e}_x$ is in the straight ahead direction in the horizontal plane, $\mathbf{e}_y$ is in the lateral direction and $\mathbf{e}_z$ points upwards in the vertical direction. $\mathbf{e}_x$ and $\mathbf{e}_z$ thus define the mid-sagittal plane; $\mathbf{e}_y$ and $\mathbf{e}_z$ define the coronal plane. The principal axes of rotations $(L_x, L_y, L_z)$ are set parallel to the head fixed co-ordinate system. A reference eye orientation called the primary position is chosen with the gaze direction being $(1, 0, 0)$ in the above co-ordinates. How is Listing's law expressed in terms of the Lie algebra of $SO(3)$? The allowed positions are generated by linear combinations of $L_z$ and $L_y$ only. This 2D subspace of the Lie algebra,

$$l = Span\{L_y, L_z\}, \tag{14}$$

is Listing's plane. Denoting $Span\{L_x\}$ by $h$, we have a decomposition of the Lie algebra $so(3)$ into a direct sum of two linear subspaces:

$$g = l \oplus h. \tag{15}$$

Every vector $v \in g$ can be projected onto its component which is in $l$:

$$v = v_l + v_h \xrightarrow{proj.} v_l. \tag{16}$$

Until now, only the linear structure has been considered. In addition, $h$ is closed under the bracket operation:

$$[L_x, L_x] = 0 \ \in h, \tag{17}$$

and because $h$ is closed both under vector addition and the Lie bracket, it is a subalgebra of $g$. In contrast, $l$ is not a subalgebra because it is not closed under commutation (8). The fact that $h$ stands as an algebra on its own implies that it has a corresponding group $H$, just as $g = so(3)$ corresponds to $G = SO(3)$. The subalgebra $h$ generates rotations about the $x$ axis, and therefore $H$ is $SO(2)$, the group of rotations in a plane.

The group $G = SO(3)$ does not have a linear structure. We may still ask whether some kind of decomposition and projection can be achieved in $G$ in analogy to (15,16). The answer is positive and the projection is performed as follows: take any element of the group, $a \in G$, and multiply it by all the elements of the subgroup $H$. This gives a subset in $G$ which is considered as a single object $\tilde{a}$ called a *coset*:

$$\tilde{a} = \{ab \mid b \in H\}. \tag{18}$$

The set of all cosets constitutes the *quotient space*. It is written as

$$S \equiv G/H = SO(3)/SO(2) \tag{19}$$

because mapping the group to the quotient space can be understood as dividing $G$ by $H$. The quotient space is not a group, and this corresponds to the fact that the subspace $l$ above (14) is not a subalgebra. The quotient space has been constructed algebraically but is difficult to visualize; however, it is mathematically equivalent

Table 1: Summary table of biological notions and the corresponding mathematical representation, both in terms of the rotation group and its Lie algebra.

| Biological notion | Lie Algebra | Rotation Group |
|---|---|---|
| general eye position | $g = so(3) = h \oplus l$ | $G = SO(3)$ |
| primary position | $0_g \in g$ | $e \in G$ |
| eye torsion | $h = Span\{L_x\}$ | $H = SO(2)$ |
| "allowed" eye positions | $l = Span\{L_y, L_z\}$ (Listing's plane) | $S = G/H = SO(3)/SO(2)$ $\cong S^2$ (Donders' sphere of gaze directions) |

to another space — the unit sphere $S^2$ (embedded in $\mathbb{R}^3$). This equivalence can be seen in the following way: a unit vector in $\mathbb{R}^3$, e.g. $e = (1,0,0)$, can be rotated so that its head reaches every point on the unit sphere $S^2$; however, for any such point there are infinitely many rotations by which the point can be reached. Moreover, all the rotations around the $x$ axis leave the vector e above invariant. We therefore have to "factor out" these rotations (of $H = SO(2)$) in order to eliminate the above degeneracy and to obtain a one-to-one correspondence between the required subset of rotations and the sphere. This is achieved by going to the quotient space.

The matrix of a torsionless rotation (generated by elements in Listing's plane) is obtained by setting $\theta_x = 0$ in (9):

$$R = \begin{pmatrix} \cos\theta & \sin\theta\sin\phi & \sin\theta\cos\phi \\ -\sin\theta\sin\phi & \cos\theta + (1-\cos\theta)\cos^2\phi & \cos\phi\sin\phi(1-\cos\theta) \\ -\sin\theta\cos\phi & \cos\phi\sin\phi(1-\cos\theta) & \cos\theta + (1-\cos\theta)\sin^2\phi \end{pmatrix}, \quad (20)$$

where $\theta = \sqrt{\theta_y^2 + \theta_z^2}$ is the total angle of rotation and $\phi$ is the angle between $\theta$ and the $y$ axis in the $\theta_y - \theta_z$ plane, i.e. $(\theta, \phi)$ are polar co-ordinates in Listing's plane. Notice that the first column on the left constitutes the Cartesian co-ordinates of a point on a sphere of unit radius (Gilmore, 1974).

As we have just seen, there is an exact correspondence between the group level and the Lie algebra level. In fact, the two describe the same reality, the former in a global manner and the latter in an infinitesimal one. Table 1 summarizes the important biological notions concerning Listing's law together with their corresponding mathematical representations. The connection between Donders' law and Listing's law can now be seen in a clear and intuitive way. The sphere, which was obtained by eliminating torsion, is the space of gaze directions. Recall that Donders' law states that the orientation of the eye is determined uniquely by its gaze direction. Listing's law implies that we need only take into consideration the gaze direction and disregard torsion. In order to emphasize this point, we use the fact that *locally*, $SO(3)$ looks like a product of topological spaces: [3]

$$P = U \times SO(2) \qquad \text{where} \qquad U \subset S^2. \qquad (21)$$

$U$ parameterizes gaze direction and $SO(2)$ — torsion. Donders' law restricts eye orientation to an unknown 2D submanifold of the product space $P$. Listing's law shows that the submanifold is $U$, a piece of the sphere. This representation is advantageous for biological modelling, because it mathematically sets apart the degrees of freedom of gaze orientation from torsion, which also differ functionally.

## 4  AXES OF ROTATION FOR LISTING'S LAW

As mentioned in the introduction, moving between two (non-primary) positions requires a rotation whose axis (i.e. angular velocity vector) lies outside Listing's plane. This is a result of the group structure of $SO(3)$. Had the axis of rotation been contained within Listing's plane, the matrices of the quotient space (20) should have been closed under multiplication so as to form a subgroup of $SO(3)$. In other words, if $r_i$ and $r_f$ are matrices representing the current and target orientations of the eye corresponding to axes in Listing's plane, then $r_f \cdot r_i^{-1}$ should have been a matrix of the same form (20); however, as explained in Section 3, this condition is not fulfilled.

Finally, since normal saccades involve rotations about a single axis, they are one-parameter subgroups generated by a single element of the Lie algebra (1). In addition, they have the property of being geodesic curves in the group manifold under the natural metric which is given by the bilinear Cartan-Killing form of the group (Choquet-Bruhat et al., 1982).

## 5  CONCLUSION

We have analysed the geometry of eye rotations using basic Lie group theory and differential geometry. The unifying view presented here can serve to improve the understanding of the oculomotor system. It may also be extended to study the three dimensional rotations of the joints of the upper limb.

### Acknowledgements

We would like to thank Stephen Gelbart, Dragana Todorić and Yosef Yomdin for instructive conversations on the mathematical background and Dario Liebermann for fruitful discussions. Special thanks go to Stan Gielen for conversations which initiated this work.

## Footnotes

*hand@wisdom.weizmann.ac.il

[1]In contrast to this third order expansion, second order approximations usually appear in the literature; see for example equation B2 in (Van Rijn & Van Den Berg, 1993).

[2] Geometrically, each point $q \in \mathcal{S}^3$ can be connected to the center of the sphere by a line. Another line runs from $q_e$ in the direction parallel to the vector part of $q$ within the tangent space. The intersection of the two lines is the projected point. Numerically, one simply takes the vector part of $q$ divided by its scalar part.

[3] $SO(3)$ is a principal bundle over $S^2$ with fiber $SO(2)$.

### References

Choquet-Bruhat Y., De Witt-Morette C. & Dillard-Bleick M., *Analysis, Manifolds and Physics*, North-Holland (1982).

Gilmore R., *LieGroups, Lie Algebras, and Some of Their Applications*, Wiley (1974).

Hepp K., *Commun. Math. Phys.* **132** (1990) 285-292.

Hestenes D., *Neural Networks* **7**, No. 1 (1994) 65-77.

Minken A.W.H. Van Opstal A.J. & Van Gisbergen J.A.M., *Exp. Brain Research* **93** (1993) 521-533.

Tweed, D. & Vilis T., *J. Neurophysiology* **58** (1987) 832-849.

Tweed D. & Vilis T., *Vision Research* **30** (1990) 111-127.

Van Opstal J., "Representations of Eye Positions in Three Dimensions", in *Multisensory Control of Movement*, ed. Berthoz A., (1993) 27-41.

Van Rijn L.J. & Van Den Berg A.V., *Vision Research* **33**, No. 5/6 (1993) 691-708.

Varadarajan V.S., *Lie Groups, Lie Algebras, and Their Reps.*, Prentice-Hall (1974).

Westheimer G., *Journal of the Optical Society of America* **47** (1957) 967-974.